# A Fast, Consistent Kernel Two-Sample Test

**Arthur Gretton**
Carnegie Mellon University
MPI for Biological Cybernetics
*arthur.gretton@gmail.com*

**Kenji Fukumizu**
Inst. of Statistical Mathematics
Tokyo Japan
*fukumizu@ism.ac.jp*

**Zaid Harchaoui**
Carnegie Mellon University
Pittsburgh, PA, USA
*zaid.harchaoui@gmail.com*

**Bharath K. Sriperumbudur**
Dept. of ECE, UCSD
La Jolla, CA 92037
*bharathsv@ucsd.edu*

## Abstract

A kernel embedding of probability distributions into reproducing kernel Hilbert spaces (RKHS) has recently been proposed, which allows the comparison of two probability measures $P$ and $Q$ based on the distance between their respective embeddings: for a sufficiently rich RKHS, this distance is zero if and only if $P$ and $Q$ coincide. In using this distance as a statistic for a test of whether two samples are from different distributions, a major difficulty arises in computing the significance threshold, since the empirical statistic has as its null distribution (where $P = Q$) an infinite weighted sum of $\chi^2$ random variables. Prior finite sample approximations to the null distribution include using bootstrap resampling, which yields a consistent estimate but is computationally costly; and fitting a parametric model with the low order moments of the test statistic, which can work well in practice but has no consistency or accuracy guarantees. The main result of the present work is a novel estimate of the null distribution, computed from the eigenspectrum of the Gram matrix on the aggregate sample from $P$ and $Q$, and having lower computational cost than the bootstrap. A proof of consistency of this estimate is provided. The performance of the null distribution estimate is compared with the bootstrap and parametric approaches on an artificial example, high dimensional multivariate data, and text.

## 1 Introduction

Learning algorithms based on kernel methods have enjoyed considerable success in a wide range of supervised learning tasks, such as regression and classification [25]. One reason for the popularity of these approaches is that they solve difficult non-parametric problems by representing the data points in high dimensional spaces of features, specifically reproducing kernel Hilbert spaces (RKHSs), in which linear algorithms can be brought to bear. While classical kernel methods have addressed the mapping of individual points to feature space, more recent developments [14, 29, 28] have focused on the embedding of probability distributions in RKHSs. When the embedding is injective, the RKHS is said to be *characteristic* [11, 29, 12], and the distance between feature mappings constitutes a metric on distributions. This distance is known as the maximum mean discrepancy (MMD).

One well-defined application of the MMD is in testing whether two samples are drawn from two different distributions (i.e., a two-sample or homogeneity test). For instance, we might wish to find whether DNA microarrays obtained on the same tissue type by different labs are distributed identically, or whether differences in lab procedure are such that the data have dissimilar distributions (and cannot be aggregated) [8]. Other applications include schema matching in databases, where tests of distribution similarity can be used to determine which fields correspond [14], and speaker

verification, where MMD can be used to identify whether a speech sample corresponds to a person for whom previously recorded speech is available [18].

A major challenge when using the MMD in two-sample testing is in obtaining a significance threshold, which the MMD should exceed with small probability when the null hypothesis (that the samples share the same generating distribution) is satisfied. Following [14, Section 4], we define this threshold as an upper quantile of the asymptotic distribution of the MMD under the null hypothesis. Unfortunately this null distribution takes the form of an infinite weighted sum of $\chi^2$ random variables. Thus, obtaining a *consistent* finite sample estimate of this threshold — that is, an estimate that converges to the true threshold in the infinite sample limit — is a significant challenge. Three approaches have previously been applied: distribution-free large deviation bounds [14, Section 3], which are generally too loose for practical settings; fitting to the Pearson family of densities [14], a simple heuristic that performs well in practice, but has no guarantees of accuracy or consistency; and a bootstrap approach, which is guaranteed to be consistent, but has a high computational cost.

The main contribution of the present study is a consistent finite sample estimate of the null distribution (not based on bootstrap), and a proof that this estimate converges to the true null distribution in the infinite sample limit. Briefly, the infinite sequence of weights that defines the null distribution is identical to the sequence of normalized eigenvalues obtained in kernel PCA [26, 27, 7]. Thus, we show that the null distribution defined using finite sample estimates of these eigenvalues converges to the population distribution, using only convergence results on certain statistics of the eigenvalues. In experiments, our new estimate of the test threshold has a smaller computational cost than that of resampling-based approaches such as the bootstrap, while providing performance as good as the alternatives for larger sample sizes.

We begin our presentation in Section 2 by describing how probability distributions may be embedded in an RKHS. We also review the maximum mean discrepancy as our chosen distance measure on these embeddings, and recall the asymptotic behaviour of its finite sample estimate. In Section 3, we present both moment-based approximations to the null distribution of the MMD (which have no consistency guarantees); and our novel, consistent estimate of the null distribution, based on the spectrum of the kernel matrix over the aggregate sample. Our experiments in Section 4 compare the different approaches on an artificial dataset, and on high-dimensional microarray and neuroscience data. We also demonstrate the generality of a kernel-based approach by testing whether two samples of text are on the same topic, or on different topics.

## 2   Background

In testing whether two samples are generated from the same distribution, we require both a measure of distance between probabilities, and a notion of whether this distance is statistically significant. For the former, we define an embedding of probability distributions in a reproducing kernel Hilbert space (RKHS), such that the distance between these embeddings is our test statistic. For the latter, we give an expression for the asymptotic distribution of this distance measure, from which a significance threshold may be obtained.

Let $\mathcal{F}$ be an RKHS on the separable metric space $\mathcal{X}$, with a continuous feature mapping $\phi(x) \in \mathcal{F}$ for each $x \in \mathcal{X}$. The inner product between feature mappings is given by the positive definite kernel function $k(x, x') := \langle \phi(x), \phi(x') \rangle_{\mathcal{F}}$. We assume in the following that the kernel $k$ is bounded. Let $\mathcal{P}$ be the set of Borel probability measures on $\mathcal{X}$. Following [4, 10, 14], we define the mapping to $\mathcal{F}$ of $P \in \mathcal{P}$ as the expectation of $\phi(x)$ with respect to $P$, or

$$
\begin{aligned}
\mu_P : \mathcal{P} &\rightarrow \mathcal{F} \\
P &\mapsto \int_{\mathcal{X}} \phi(x) dP.
\end{aligned}
$$

The maximum mean discrepancy (MMD) [14, Lemma 7] is defined as the distance between two such mappings,

$$
\begin{aligned}
\mathrm{MMD}(P, Q) &:= \|\mu_P - \mu_Q\|_{\mathcal{F}} \\
&= \left( \mathbf{E}_{x, x'}(k(x, x')) + \mathbf{E}_{y, y'} k(y, y') - 2\mathbf{E}_{x, y} k(x, y) \right)^{1/2},
\end{aligned}
$$

where $x$ and $x'$ are independent random variables drawn according to $P$, $y$ and $y'$ are independent and drawn according to $Q$, and $x$ is independent of $y$. This quantity is a *pseudo-metric* on distributions: that is, it satisfies all the qualities of a metric besides $\mathrm{MMD}(P, Q) = 0$ iff $P = Q$. For MMD

to be a metric, we require that the kernel be characteristic [11, 29, 12].[1] This criterion is satisfied for many common kernels, such as the Gaussian kernel (both on compact domains and on $\mathbb{R}^d$) and the $B_{2l+1}$ spline kernel on $\mathbb{R}^d$.

We now consider two possible empirical estimates of the MMD, based on i.i.d. samples $(x_1, \ldots, x_m)$ from $P$ and $(y_1, \ldots, y_m)$ from $Q$ (we assume an equal number of samples for simplicity). An *unbiased* estimate of MMD is the one-sample U-statistic

$$\mathrm{MMD}_u^2 := \frac{1}{m(m-1)} \sum_{i \neq j}^m h(z_i, z_j), \tag{1}$$

where $z_i := (x_i, y_i)$ and $h(z_i, z_j) := k(x_i, x_j) + k(y_i, y_j) - k(x_i, y_j) - k(x_j, y_i)$. We also define the *biased* estimate $\mathrm{MMD}_b^2$ by replacing the U-statistic in (1) with a V-statistic (the sum then includes terms $i = j$).

Our goal is to determine whether $P$ and $Q$ differ, based on $m$ samples from each. To this end, we require a measure of whether $\mathrm{MMD}_u^2$ differs significantly from zero; or, if the biased statistic $\mathrm{MMD}_b^2$ is used, whether this value is significantly greater than its expectation when $P = Q$. In other words we conduct a hypothesis test with null hypothesis $\mathcal{H}_0$ defined as $P = Q$, and alternative hypothesis $\mathcal{H}_1$ as $P \neq Q$. We must therefore specify a threshold that the empirical MMD will exceed with small probability, when $P = Q$. For an asymptotic false alarm probability (Type I error) of $\alpha$, an appropriate threshold is the $1 - \alpha$ quantile of the asymptotic distribution of the empirical MMD assuming $P = Q$. According to [14, Theorem 8], this distribution takes the form

$$m\mathrm{MMD}_u^2 \xrightarrow[D]{} \sum_{l=1}^{\infty} \lambda_l (z_l^2 - 2), \tag{2}$$

where $\xrightarrow[D]{}$ denotes convergence in distribution, $z_l \sim \mathcal{N}(0, 2)$ i.i.d., $\lambda_i$ are the solutions to the eigenvalue equation

$$\int_{\mathcal{X}} \tilde{k}(x_i, x_j) \psi_l(x_i) dP := \lambda_l \psi_l(x_j), \tag{3}$$

and $\tilde{k}(x_i, x_j) := k(x_i, x_j) - \mathbf{E}_x k(x_i, x) - \mathbf{E}_x k(x, x_i) + \mathbf{E}_{x,x'} k(x, x')$. Consistency in power of the resulting hypothesis test (that is, the convergence of its Type II error to zero for increasing $m$) is shown in [14].

The eigenvalue problem (3) has been studied extensively in the context of kernel PCA [26, 27, 7]: this connection will be used in obtaining a finite sample estimate of the null distribution in (2), and we summarize certain important results. Following [3, 10], we define the covariance operator $C : \mathcal{F} \to \mathcal{F}$ as

$$\begin{aligned} \langle f, Cf \rangle_{\mathcal{F}} & := \mathrm{var}(f(x)) \\ & = \mathbf{E}_x f^2(x) - [\mathbf{E}_x f(x)]^2 . \end{aligned} \tag{4}$$

The eigenvalues $\lambda_l$ of $C$ are the solutions to the eigenvalue problem in (3) [19, Proposition 2]. Following e.g. [27, p.2511], empirical estimates of these eigenvalues are

$$\hat{\lambda}_l = \frac{1}{m} \nu_l \tag{5}$$

where $\nu_l$ are the eigenvalues of the centered Gram matrix

$$\widetilde{K} := HKH,$$

$K_{i,j} := k(x_i, x_j)$, and $H = I - \frac{1}{m} 11^\top$ is a centering matrix. Finally, by subtracting $m\mathrm{MMD}_u^2$ from $m\mathrm{MMD}_b^2$, we observe that these differ by a quantity with expectation $\mathrm{tr}(C) = \sum_{l=1}^{\infty} \lambda_l$, and thus

$$m\mathrm{MMD}_b^2 \xrightarrow[D]{} \sum_{l=1}^{\infty} \lambda_l z_l^2 .$$

# 3 Theory

In the present section, we describe three approaches for approximating the null distribution of MMD. We first present the Pearson curve and Gamma-based approximations, which consist of parametrized families of distributions that we fit by matching the low order moments of the empirical MMD. Such approximations can be accurate in practice, although they remain heuristics with no consistency guarantees. Second, we describe a null distribution estimate based on substituting the empirical estimates (5) of the eigenvalues into (2). We prove that this estimate converges to its population counterpart in the large sample limit.

## 3.1 Moment-based null distribution estimates

The Pearson curves and the Gamma approximation are both based on the low order moments of the empirical MMD. The second and third moments for MMD are obtained in [14]:

$$\mathbf{E}\left(\left[\mathrm{MMD}_u^2\right]^2\right) = \frac{2}{m(m-1)} \mathbf{E}_{z,z'}\left[h^2(z,z')\right] \text{ and} \tag{6}$$

$$\mathbf{E}\left(\left[\mathrm{MMD}_u^2\right]^3\right) = \frac{8(m-2)}{m^2(m-1)^2} \mathbf{E}_{z,z'}\left[h(z,z')\mathbf{E}_{z''}\left(h(z,z'')h(z',z'')\right)\right] + O(m^{-4}). \tag{7}$$

Pearson curves take as arguments the variance, skewness and kurtosis As in [14], we replace the kurtosis with a lower bound due to [31], $\mathrm{kurt}\left(\mathrm{MMD}_u^2\right) \geq \left(\mathrm{skew}\left(\mathrm{MMD}_u^2\right)\right)^2 + 1$. An alternative, more computationally efficient approach is to use a two-parameter Gamma approximation [20, p. 343, p. 359],

$$m\mathrm{MMD}_b(Z) \sim \frac{x^{\alpha-1}e^{-x/\beta}}{\beta^\alpha\Gamma(\alpha)} \quad \text{where} \quad \alpha = \frac{(\mathbf{E}(\mathrm{MMD}_b(Z)))^2}{\mathrm{var}(\mathrm{MMD}_b(Z))}, \quad \beta = \frac{m\mathrm{var}(\mathrm{MMD}_b(Z))}{\mathbf{E}(\mathrm{MMD}_b(Z))}, \tag{8}$$

and we use the *biased* statistic $\mathrm{MMD}_b^2$. Although the Gamma approximation is necessarily less accurate than the Pearson approach, it has a substantially lower computational cost ($O(m^2)$ for the Gamma approximation, as opposed to $O(m^3)$ for Pearson). Moreover, we will observe in our experiments that it performs remarkably well, at a substantial cost saving over the Pearson curves.

## 3.2 Null distribution estimates using Gram matrix spectrum

In [14, Theorem 8], it was established that for large sample sizes, the null distribution of MMD approaches an infinite weighted sum of independent $\chi_1^2$ random variables, the weights being the population eigenvalues of the covariance operator $C$. Hence, an efficient and theoretically grounded way to calibrate the test is to compute the quantiles by replacing the population eigenvalues of $C$ with their empirical counterparts, as computed from the Gram matrix (see also [18], where a similar strategy is proposed for the KFDA test with fixed regularization).

The following result shows that this empirical estimate of the null distribution converges in distribution to its population counterpart. In other words, a test using the MMD statistic, with the threshold computed from quantiles of the null distribution estimate, is *asymptotically consistent in level*.

**Theorem 1** *Let $z_1, \ldots, z_l, \ldots$ be an infinite sequence of i.i.d. random variables, with $z_1 \sim \mathcal{N}(0,2)$. Assume $\sum_{l=1}^\infty \lambda_l^{1/2} < \infty$. Then, as $m \to \infty$*

$$\sum_{l=1}^\infty \hat{\lambda}_l(z_l^2 - 2) \xrightarrow{D} \sum_{l=1}^\infty \lambda_l(z_l^2 - 2).$$

*Furthermore, as $m \to \infty$*

$$\sup_t \left|\mathbf{P}\left(m\mathit{MMD}_u^2 > t\right) - \mathbf{P}\left(\sum_{l=1}^\infty \hat{\lambda}_l(z_l^2 - 2) > t\right)\right| \to 0.$$

**Proof** *(sketch)* We begin with a proof of conditions under which the sum $\sum_{l=1}^{\infty} \lambda_l(z_l^2 - 2)$ is finite w.p. 1. According to [16, Exercise 30, p. 358], we may use Kolmogorov's inequality to determine that this sum converges a.s. if

$$\sum_{l=1}^{\infty} \mathbf{E}_z[\lambda_l^2(z_l^2 - 2)^2] < \infty,$$

from which it follows that the covariance operator must be Hilbert-Schmidt: this is guaranteed by the assumption $\sum_{l=1}^{\infty} \lambda_l^{1/2} < \infty$ (see also [7]). We now proceed to the convergence result. Let $C$ and $\widehat{C}$ be the covariance operator and its empirical estimator. Let $\lambda_l$ and $\widehat{\lambda}_l$ ($l = 1, 2, \ldots$) be the eigenvalues of $C$ and $\widehat{C}$, respectively, in descending order. We want to prove

$$\sum_{p=1}^{\infty} (\widehat{\lambda}_l - \lambda_l) Z_l^2 \quad \rightarrow \quad 0 \tag{9}$$

in probability as $n \rightarrow \infty$, where $Z_p \sim N(0, 2)$ are i.i.d. random variables. The constant $-2$ in $Z_p^2 - 2$ can be neglected as $\mathrm{Tr}[\widehat{C}] \rightarrow \mathrm{Tr}[C]$, where the proof is given in the online supplement. Thus

$$\left| \sum_l (\widehat{\lambda}_l - \lambda_l) Z_l^2 \right| \leq \left| \sum_l \widehat{\lambda}_l^{1/2} (\widehat{\lambda}_l^{1/2} - \lambda_l^{1/2}) Z_l^2 \right| + \left| \sum_l (\widehat{\lambda}_l^{1/2} - \lambda_l^{1/2}) \lambda_l^{1/2} Z_l^2 \right|$$

$$\leq \left\{ \sum_l \widehat{\lambda}_l Z_l^4 \right\}^{1/2} \left\{ \sum_l |\widehat{\lambda}_l^{1/2} - \lambda_l^{1/2}|^2 \right\}^{1/2}$$

$$+ \left\{ \sum_l \lambda_l Z_l^4 \right\}^{1/2} \left\{ \sum_l |\widehat{\lambda}_l^{1/2} - \lambda_l^{1/2}|^2 \right\}^{1/2} \quad \text{(Cauchy-Schwarz)}. \tag{10}$$

We now establish $\sum_l \lambda_l Z_l^4$ and $\sum_l \widehat{\lambda}_l Z_l^4$ are of $O_p(1)$. The former follows from Chebyshev's inequality. To prove the latter, we use that since $\widehat{\lambda}_i$ and $Z_i$ are independent,

$$\mathbf{E} \sum_i \widehat{\lambda}_i Z_i^4 = \sum_i \mathbf{E}[\widehat{\lambda}_i] \mathbf{E}[Z_i^4] = \kappa \mathbf{E}[tr(\widehat{C})], \tag{11}$$

where $\kappa = \mathbf{E}[Z^4]$. Since $\mathbf{E}[tr(\widehat{C})]$ is bounded when the kernel has bounded expectation, we again have the desired result by Chebyshev's inequality. The proof is complete if we show

$$\sum_l (\widehat{\lambda}_l^{1/2} - \lambda_l^{1/2})^2 = o_p(1). \tag{12}$$

From

$$\left| \widehat{\lambda}_l^{1/2} - \lambda_l^{1/2} \right|^2 \leq \left| \widehat{\lambda}_l^{1/2} - \lambda_l^{1/2} \right| (\widehat{\lambda}_l^{1/2} + \lambda_l^{1/2}) = \left| \widehat{\lambda}_l - \lambda_l \right|, \tag{13}$$

we have

$$\sum_l \left| \widehat{\lambda}_l^{1/2} - \lambda_l^{1/2} \right|^2 \leq \sum_l |\widehat{\lambda}_l - \lambda_l|.$$

It is known as an extension of the Hoffmann-Wielandt inequality that

$$\sum_l \left| \widehat{\lambda}_l - \lambda_l \right| \leq \|\widehat{C} - C\|_1,$$

where $\| \cdot \|_1$ is the trace norm (see [23], also shown in [5, p. 490]). Using [18, Prop. 12], which gives $\|\widehat{C} - C\|_1 \rightarrow 0$ in probability, the proof of the first statement is completed. The proof of the second statement follows immediately from the Polya theorem [21], as in [18]. ■

## 3.3 Discussion

We now have several ways to calibrate the MMD test statistic, ranked in order of increasing computational cost: 1) the Gamma approximation, 2) the "empirical null distribution": that is, the null distribution estimate using the empirical Gram matrix spectrum, and 3) the Pearson curves, and

the resampling procedures (subsampling or bootstrap with replacement). We include the final two approaches in the same cost category since even though the Pearson approach scales worse with $m$ than the bootstrap ($O(m^3)$ vs $O(m^2)$), the bootstrap has a higher cost for sample sizes less than about $10^3$ due the requirement to repeatedly re-compute the test statistic. We also note that our result of large-sample consistency in level holds under a restrictive condition on the decay of the spectrum of the covariance operator, whereas the Gamma approximation calculations are straightforward and remain possible for any spectrum decay behaviour. The Gamma approximation remains a heuristic, however, and we give an example of a distribution and kernel for which it performs less accurately than the spectrum-based estimate in the upper tail, which is of most interest for testing purposes.

## 4 Experiments

In this section, we compare the four approaches to obtaining the null distribution, both in terms of the approximation error computed with respect to simulations from the true null, and when used in homogeneity testing. Our approaches are denoted *Gamma* (the two-parameter Gamma approximation), *Pears* (the Pearson curves based on the first three moments, using a lower bound for the kurtosis), *Spec* (our new approximation to the null distribution, using the Gram matrix eigenspectrum), and *Boot* (the bootstrap approach).

**Artificial data:** We first provide an example of a distribution $P$ for which the heuristics *Gamma* and *Pears* have difficulty in approximating the null distribution, whereas *Spec* converges. We chose $P$ to be a mixture of normals $P = 0.5 * \mathcal{N}(-1, 0.44) + 0.5 * \mathcal{N}(+1, 0.44)$, and $k$ as a Gaussian kernel with bandwidth ranging over $\sigma = 2^{-4}, 2^{-3}, 2^{-2}, 2^{-1}, 2^0, 2^1, 2^2$. The sample sizes were set to $m = 5000$, the total sample size hence being $10,000$, and the results were averaged over $50,000$ replications. The eigenvalues of the Gram matrix were estimated in this experiment using [13], which is slower but more accurate than standard Matlab routines. The true quantiles of the MMD null distribution, referred to as the oracle quantiles, were estimated by Monte Carlo simulations with $50,000$ runs. We report the empirical performance of *Spec* compared to the oracle in terms of $\Delta_q = \max_{t_r:q<r<1} |\mathbf{P}(mMMD_u^2 > t_r) - \widehat{\mathbf{P}}_m(mMMD_u^2 > t_r)|$, where $t_q$ is such that $\mathbf{P}(mMMD_u^2 > t_q) = q$ for $q = 0.6, 0.7, 0.8, 0.9$, and $\widehat{\mathbf{P}}_m$ is the *Spec* null distribution estimate obtained with $m$ samples from each of $P$ and $Q$. We also use this performance measure for the *Gamma* and *Pears* approximations. This focuses the performance comparison on the quantiles corresponding to the upper tail of the null distribution, while still addressing uniform accuracy over a range of thresholds so as to ensure reliable $p$-values. The results are shown in Figure 1, and demonstrate that for this combination of distribution and kernel, *Spec* performs almost uniformly better than both *Gamma* and *Pears*. We emphasize that the performance advantage of *Spec* is greatest when we restrict ourselves to higher quantiles, which are of most interest in testing.

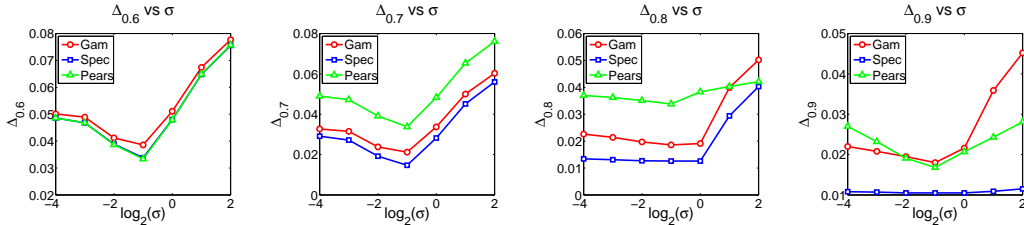

Figure 1: Evolution of $\Delta_q$ for resp. the Gamma (Gam), Spectrum (Spec), and Pearson (Pears) approximations to the null distribution, as the Gaussian kernel bandwidth parameter varies. From left to right, plots of $\Delta_q$ versus $\sigma = 2^{-4}, 2^{-3}, \dots, 2^2$ for $q = 0.6, 0.7, 0.8, 0.9$.

**Benchmark data:** We next demonstrate the performance of the MMD tests on a number of multivariate datasets, taken from [14, Table 1]. We compared microarray data from normal and tumor tissues (Health status), microarray data from different subtypes of cancer (Subtype), and local field potential (LFP) electrode recordings from the Macaque primary visual cortex (V1) with and without spike events (Neural Data I and II, described in [24]). In all cases, we were provided with two samples having different statistical properties, where the detection of these differences was made difficult by the high data dimensionality (for the microarray data, density estimation is impossi-

ble given the small sample size and high data dimensionality, and a successful test cannot rely on accurate density estimates as an intermediate step).

In computing the null distributions for both the *Spec* and *Pears* cases, we drew 500 samples from the associated null distribution estimates, and computed the test thresholds using the resulting empirical quantiles. For the *Spec* case, we computed the eigenspectrum on the gram matrix of the aggregate data from $P$ and $Q$, retaining in all circumstances the maximum number $2m - 1$ of nonzero eigenvalues of the empirical Gram matrix. This is a conservative approach, given that the Gram matrix spectrum may decay rapidly [2, Appendix C], in which case it might be possible to safely discard the smallest eigenvalues. For the bootstrap approach *Boot*, we aggregated points from the two samples, then assigned these randomly without replacement to $P$ and $Q$. In our experiments, we performed 500 such iterations, and used the resulting histogram of MMD values as our null distribution. We used a Gaussian kernel in all cases, with the bandwidth set to the median distance between points in the aggregation of samples from $P$ and $Q$.

We applied our tests to the benchmark data as follows: Given datasets A and B, we either drew one sample with replacement from A and the other from B (in which case a Type II error was made when the null hypothesis $\mathcal{H}_0$ was accepted); or we drew both samples with replacement from a single pool consisting of A and B combined (in which case a Type I error was made when $\mathcal{H}_0$ was rejected: this should happen a fraction $1 - \alpha$ of the time). This procedure was repeated 1000 times to obtain average performance figures. We summarize our results in Table 1. Note that an extensive benchmark of the MMD *Boot* and *Pears* tests against other nonparametric approaches to two-sample testing is provided in [14]: these include the the Friedman-Rafsky generalisation of the Kolmogorov-Smirnov and Wald-Wolfowitz tests [9], the Biau-Györfi test [6], and the Hall-Tajvidi test [17]. See [14] for details.

We observe that the kernel tests perform extremely well on these data: the Type I error is in the great majority of cases close to its design value of $1 - \alpha$, and the Type II error is very low (and often zero). The *Spec* test is occasionally slightly conservative, and has a lower Type I error than required: this is most pronounced in the Health Status dataset, for which the sample size $m$ is low. The computational cost shows the expected trend, with *Gamma* being least costly, followed by *Spec*, *Pears*, and finally *Boot* (this trend is only visible for the larger $m = 500$ datasets). Note that for yet larger sample sizes, however, we expect the cost of *Pears* to exceed that of the remaining methods, due to its $O(m^3)$ cost requirement (vs $O(m^2)$ for the other approaches).

| Dataset | Attribute | Gamma | Pears | Spec | Boot |
|---|---|---|---|---|---|
| Neural Data I | Type I/Type II | 0.95 / 0.00 | 0.96 / 0.00 | 0.96 / 0.00 | 0.96 / 0.00 |
| | Time (sec) | 0.06 | 3.92 | 2.79 | 5.79 |
| Neural Data II | Type I/Type II | 0.96 / 0.00 | 0.96 / 0.00 | 0.97 / 0.00 | 0.96 / 0.00 |
| | Time (sec) | 0.08 | 3.97 | 2.91 | 8.08 |
| Health status | Type I/Type II | 0.96 / 0.00 | 0.96 / 0.00 | 0.98 / 0.00 | 0.95 / 0.00 |
| | Time (sec) | 0.01 | 0.01 | 0.01 | 0.03 |
| Subtype | Type I/Type II | 0.95 / 0.02 | 0.95 / 0.01 | 0.96 / 0.01 | 0.94 / 0.01 |
| | Time (sec) | 0.05 | 0.05 | 0.05 | 0.07 |

Table 1: Benchmarks for the kernel two-sample tests on high dimensional multivariate data. Type I and Type II errors are provided, as are average run times. Sample size (dimension): Neural I 500 (63) ; Neural II 500 (100); Health Status 25 (12,600); Subtype 25 (2,118).

Finally, we demonstrate the performance of the test on structured (text) data. Our data are taken from the Canadian Hansard corpus ($\mathtt{http://www.isi.edu/natural-language/download/hansard/}$). As in the earlier work on dependence testing presented in [15], debate transcripts on the three topics of agriculture, fisheries, and immigration were used. Transcripts were in English and French, however we confine ourselves to reporting results on the English data (the results on the French data were similar). Our goal was to distinguish samples on different *topics*, for instance $P$ being drawn from transcripts on agriculture and $Q$ from transcripts on immigration (in the null case, both samples were from the same topic). The data were processed following the same procedures as in [15]. We investigated two different kernels on text: the $k$-substring kernel of [22, 30] with $k = 10$, and a bag-of-words kernel. In both cases, we computed kernels between five-line extracts, ignoring lines shorter than five words long. Results are presented in Figure 2, and represent an average over all three combinations of

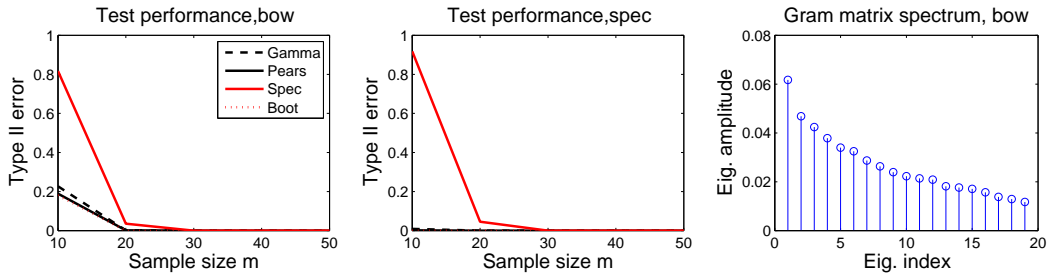

Figure 2: Canadian Hansard data. **Left:** Average Type II error over all of agriculture-fisheries, agriculture-immigration, and fisheries-immigration, for the bag-of-words kernel. **Center:** Average Type II error for the k-substring kernel. **Right:** Eigenspectrum of a centered Gram matrix obtained by drawing $m = 10$ points from each of $P$ and $Q$, where $P \neq Q$, for the bag-of-words kernel.

different topic pairs: agriculture-fisheries, agriculture-immigration, and fisheries-immigration. For each topic pairing, results are averaged over 300 repetitions.

We observe that in general, the MMD is very effective at distinguishing distributions of text fragments on different topics: for sample sizes above 30, all the test procedures are able to detect differences in distribution with zero Type II error, for both kernels. When the $k$-substring kernel is used, the Boot, Gamma, and Pears approximations can distinguish the distributions for sample sizes as low as 10: this indicates that a more sophisticated encoding of the text than provided by bag-of-words results in tests of greater sensitivity (consistent with the independence testing observations of [15]).

We now investigate the fact that for sample sizes below $m = 30$ on the Hansard data, the *Spec* test has a much higher Type II error the alternatives. The $k$-substring and bag-of-words kernels are diagonally dominant: thus for small sample sizes, the empirical estimate of the kernel spectrum is effectively truncated at a point where the eigenvalues remain large, introducing a bias (Figure 2). This effect vanishes on the Hansard benchmark once the number of samples reaches 25-30. By contrast, for the Neural data using a Gaussian kernel, this small sample bias is not observed, and the *Spec* test has equivalent Type II performance to the other three tests (see Figure 1 in the online supplement). In this case, for sample sizes of interest (i.e., where there are sufficient samples to obtain a Type II error of less than 50%), the bias in the *Spec* test due to spectral truncation is negligible. We emphasize that the speed advantage of the *Spec* test becomes important only for larger sample sizes (and the consistency guarantee is only meaningful in this regime).

## 5    Conclusion

We have presented a novel method for estimating the null distribution of the RKHS distance between probability distribution embeddings, for use in a nonparametric test of homogeneity. Unlike previous parametric heuristics based on moment matching, our new distribution estimate is consistent; moreover, it is computationally less costly than the bootstrap, which is the only alternative consistent approach. We have demonstrated in experiments that our method performs well on high dimensional multivariate data and text, as well as for distributions where the parametric heuristics show inaccuracies. We anticipate that our approach may also be generalized to kernel independence tests [15], and to homogeneity tests based on the kernel Fisher discriminant [18].

**Acknowledgments:** The ordering of the second through fourth authors is alphabetical. We thank Choon-Hui Teo for generating the Gram matrices for the text data, Malte Rasch for his assistance in the experimental evaluation, and Karsten Borgwardt for his assistance with the microarray data. A. G. was supported by grants DARPA IPTO FA8750-09-1-0141, ONR MURI N000140710747, and ARO MURI W911NF0810242. Z. H. was supported by grants from the Technical Support Working Group through funding from the Investigative Support and Forensics subgroup and NIMH 51435, and from Agence Nationale de la Recherche under contract ANR-06-BLAN-0078 KERNSIG. B. K. S. was supported by the MPI for Biological Cybernetics, NSF (grant DMS-MSPA 0625409), the Fair Isaac Corporation and the University of California MICRO program.

## Footnotes

[1] Other interpretations of the MMD are also possible, for particular kernel choices. The most closely related is the $L_2$ distance between probability density estimates [1], although this requires the kernel bandwidth to decrease with increasing sample size. See [1, 14] for more detail. Yet another interpretation is given in [32].

## References

[1] N. Anderson, P. Hall, and D. Titterington. Two-sample test statistics for measuring discrepancies between two multivariate probability density functions using kernel-based density estimates. *Journal of*

*Multivariate Analysis*, 50:41–54, 1994.

[2] F. R. Bach and M. I. Jordan. Kernel independent component analysis. *J. Mach. Learn. Res.*, 3:1–48, 2002.

[3] C. Baker. Joint measures and cross-covariance operators. *Transactions of the American Mathematical Society*, 186:273–289, 1973.

[4] A. Berlinet and C. Thomas-Agnan. *Reproducing Kernel Hilbert Spaces in Probability and Statistics*. Springer-Verlag, Berlin, 2003.

[5] Rajendra Bhatia and Ludwig Elsner. The Hoffman-Wielandt inequality in infinite dimensions. *Proceedings of Indian Academy of Science (Mathematical Sciences)*, 104(3):483–494, 1994.

[6] G. Biau and L. Gyorfi. On the asymptotic properties of a nonparametric $l_1$-test statistic of homogeneity. *IEEE Transactions on Information Theory*, 51(11):3965–3973, 2005.

[7] G. Blanchard, O. Bousquet, and L. Zwald. Statistical properties of kernel principal component analysis. *Machine Learning*, 66:259–294, 2007.

[8] K. M. Borgwardt, A. Gretton, M. J. Rasch, H.-P. Kriegel, B. Schölkopf, and A. J. Smola. Integrating structured biological data by kernel maximum mean discrepancy. *Bioinformatics (ISMB)*, 22(14):e49–e57, 2006.

[9] J. Friedman and L. Rafsky. Multivariate generalizations of the Wald-Wolfowitz and Smirnov two-sample tests. *The Annals of Statistics*, 7(4):697–717, 1979.

[10] K. Fukumizu, F. R. Bach, and M. I. Jordan. Dimensionality reduction for supervised learning with reproducing kernel Hilbert spaces. *J. Mach. Learn. Res.*, 5:73–99, 2004.

[11] K. Fukumizu, A. Gretton, X. Sun, and B. Schölkopf. Kernel measures of conditional dependence. In *NIPS 20*, pages 489–496, 2008.

[12] K. Fukumizu, B. Sriperumbudur, A. Gretton, and B. Schölkopf. Characteristic kernels on groups and semigroups. In *NIPS 21*, pages 473–480, 2009.

[13] G. Golub and Q. Ye. An inverse free preconditioned krylov subspace method for symmetric generalized eigenvalue problems. *SIAM Journal on Scientific Computing*, 24:312–334, 2002.

[14] A. Gretton, K. Borgwardt, M. Rasch, B. Schölkopf, and A. Smola. A kernel method for the two-sample-problem. In *NIPS 19*, pages 513–520, 2007.

[15] A. Gretton, K. Fukumizu, C.-H. Teo, L. Song, B. Schölkopf, and A. Smola. A kernel statistical test of independence. In *NIPS 20*, pages 585–592, 2008.

[16] G. R. Grimmet and D. R. Stirzaker. *Probability and Random Processes*. Oxford University Press, Oxford, third edition, 2001.

[17] P. Hall and N. Tajvidi. Permutation tests for equality of distributions in high-dimensional settings. *Biometrika*, 89(2):359–374, 2002.

[18] Z. Harchaoui, F. Bach, and E. Moulines. Testing for homogeneity with kernel fisher discriminant analysis. In *NIPS 20*, pages 609–616. 2008. (long version: arXiv:0804.1026v1).

[19] M. Hein and O. Bousquet. Kernels, associated structures, and generalizations. Technical Report 127, Max Planck Institute for Biological Cybernetics, 2004.

[20] N. L. Johnson, S. Kotz, and N. Balakrishnan. *Continuous Univariate Distributions. Volume 1 (Second Edition)*. John Wiley and Sons, 1994.

[21] E. Lehmann and J. Romano. *Testing Statistical Hypothesis (3rd ed.)*. Wiley, New York, 2005.

[22] C. Leslie, E. Eskin, and W. S. Noble. The spectrum kernel: A string kernel for SVM protein classification. In *Proceedings of the Pacific Symposium on Biocomputing*, pages 564–575, 2002.

[23] A. S. Markus. The eigen- and singular values of the sum and product of linear operators. *Russian Mathematical Surveys*, 19(4):93–123, 1964.

[24] M. Rasch, A. Gretton, Y. Murayama, W. Maass, and N. K. Logothetis. Predicting spiking activity from local field potentials. *Journal of Neurophysiology*, 99:1461–1476, 2008.

[25] B. Schölkopf and A. Smola. *Learning with Kernels*. MIT Press, Cambridge, MA, 2002.

[26] B. Schölkopf, A. J. Smola, and K.-R. Müller. Nonlinear component analysis as a kernel eigenvalue problem. *Neural Computation*, 10:1299–1319, 1998.

[27] J. Shawe-Taylor, C. Williams, N. Cristianini, and J. Kandola. On the eigenspectrum of the Gram matrix and the generalisation error of kernel PCA. *IEEE Trans. Inf. Theory*, 51(7):2510–2522, 2005.

[28] A. J. Smola, A. Gretton, L. Song, and B. Schölkopf. A Hilbert space embedding for distributions. In *ALT 18*, pages 13–31, 2007.

[29] B. Sriperumbudur, A. Gretton, K. Fukumizu, G. Lanckriet, and B. Schölkopf. Injective hilbert space embeddings of probability measures. In *COLT 21*, pages 111–122, 2008.

[30] C. H. Teo and S. V. N. Vishwanathan. Fast and space efficient string kernels using suffix arrays. In *ICML*, pages 929–936, 2006.

[31] J. E. Wilkins. A note on skewness and kurtosis. *Ann. Math. Stat.*, 15(3):333–335, 1944.

[32] G. Zech and B. Aslan. A multivariate two-sample test based on the concept of minimum energy. In *PHYSTAT*, pages 97–100, 2003.

